# Consistent Minimization of Clustering Objective Functions

**Ulrike von Luxburg**
Max Planck Institute for Biological Cybernetics
ulrike.luxburg@tuebingen.mpg.de

**Sébastien Bubeck**
INRIA Futurs Lille, France
sebastien.bubeck@inria.fr

**Stefanie Jegelka**
Max Planck Institute for Biological Cybernetics
stefanie.jegelka@tuebingen.mpg.de

**Michael Kaufmann**
University of Tübingen, Germany
mk@informatik.uni-tuebingen.de

## Abstract

Clustering is often formulated as a discrete optimization problem. The objective is to find, among all partitions of the data set, the best one according to some quality measure. However, in the statistical setting where we assume that the finite data set has been sampled from some underlying space, the goal is not to find the best partition of the given sample, but to approximate the true partition of the underlying space. We argue that the discrete optimization approach usually does not achieve this goal. As an alternative, we suggest the paradigm of "nearest neighbor clustering". Instead of selecting the best out of all partitions of the sample, it only considers partitions in some restricted function class. Using tools from statistical learning theory we prove that nearest neighbor clustering is statistically consistent. Moreover, its worst case complexity is polynomial by construction, and it can be implemented with small average case complexity using branch and bound.

## 1  Introduction

Clustering is the problem of discovering "meaningful" groups in given data. Many algorithms try to achieve this by minimizing a certain quality function $Q_n$, for example graph cut objective functions such as ratio cut or normalized cut, or various criteria based on some function of the within- and between-cluster similarities. The objective of clustering is then stated as a discrete optimization problem. Given a data set $\mathcal{X}_n = \{X_1, \ldots, X_n\}$ and a clustering quality function $Q_n$, the ideal clustering algorithm should take into account all possible partitions of the data set and output the one that minimizes $Q_n$. The implicit understanding is that the "best" clustering can be any partition out of the set of all possible partitions of the data set. The algorithmic challenge is to construct an algorithm which is able to find this clustering. We will call this approach the "discrete optimization approach to clustering".

If we look at clustering from the perspective of statistical learning theory we assume that the finite data set has been sampled from an underlying data space $\mathcal{X}$ according to some probability measure. The ultimate goal in this setting is not to discover the best possible partition of the data set $\mathcal{X}_n$, but to learn the "true clustering" of the underlying space. In an approach based on quality functions, this "true clustering" can be defined easily. We choose a clustering quality function $Q$ on the set of partitions of the entire data space $\mathcal{X}$, and define the true clustering $f^*$ to be the partition minimizing $Q$. In this setting, a very important property of a clustering algorithm is consistency. Denoting the clustering constructed on the finite sample by $f_n$, we require that $Q(f_n)$ converges to $Q(f^*)$ when $n \to \infty$. The most important insight of statistical learning theory is that in order to be consistent, learning algorithms have to choose their functions from some "small" function space only. To measure the size of a function space $\mathcal{F}$ one uses the quantity $N_{\mathcal{F}}(x_1, .., x_n)$ which denotes the number

of ways in which the points $x_1, \ldots, x_n$ can be partitioned by functions in $\mathcal{F}$. One can prove that in the standard setting of statistical learning theory, a necessary condition for consistency is that $\mathbb{E} \log N_{\mathcal{F}}(x_1, \ldots, x_n)/n \to 0$ (cf. Theorem 2.3 in Vapnik, 1995, Section 12.4 of Devroye et al., 1996).

Stated like this, it becomes apparent that the two viewpoints described above are not compatible with each other. While the discrete optimization approach on any given sample attempts to find the best of all (exponentially many) partitions, the statistical learning theory approach restricts the set of candidate partitions to have sub-exponential size. Hence, from the statistical learning theory perspective, an algorithm which is considered ideal in the discrete optimization setting is likely to overfit. One can construct simple examples (cf. Bubeck and von Luxburg, 2007) which show that this indeed can happen: here the partitions constructed on the finite sample do not converge to the true clustering of the data space. In practice, for most cases the discrete optimization approach cannot be performed perfectly as the corresponding optimization problem is NP hard. Instead, people resort to heuristics. One approach is to use local optimization procedures potentially ending in local minima only (this is what happens in the $k$-means algorithm). Another approach is to construct a relaxation of the original problem which can be solved efficiently (spectral clustering is an example for this). In both cases, one usually cannot guarantee how close the heuristic solution is to the global finite sample optimum. This situation is clearly unsatisfactory: for most clustering algorithms, we neither have guarantees on the finite sample behavior of the algorithm, nor on its statistical consistency in the limit.

The following alternative approach looks much more promising. Instead of attempting to solve the discrete optimization problem over the set of all partitions, and then resorting to relaxations due to the NP-hardness of this problem, we turn the tables. Directly from the outset, we only consider candidate partitions in some restricted class $\mathcal{F}_n$ containing only polynomially many functions. Then the discrete optimization problem of minimizing $Q_n$ over $\mathcal{F}_n$ is no longer NP hard – it can trivially be solved in polynomially many steps by trying all candidates in $\mathcal{F}_n$. From a theoretical point of view this approach has the advantage that the resulting clustering algorithm has the potential of being consistent. In addition, it also leads to practical benefits: rather than dealing with uncontrolled relaxations of the original problem, we restrict the function class to some small enough subset $\mathcal{F}_n$ of "reasonable" partitions. Within this subset, we then have complete control over the solution of the optimization problem and can find the global optimum. Put another way, one can also interpret this approach as some controlled way of sparsifying the NP hard optimization problem, with the positive side effect of obeying the rules of statistical learning theory.

## 2  Nearest neighbor clustering

In the following we assume that we are given a set of data points $\mathcal{X}_n = \{X_1, \ldots, X_n\}$ and pairwise distances $d_{ij} = d(X_i, X_j)$ or pairwise similarities $s_{ij} = s(X_i, X_j)$. Let $Q_n$ be the finite sample quality function to optimize on the sample. To follow the approach outlined above we have to optimize $Q_n$ over a "small" set $\mathcal{F}_n$ of partitions of $\mathcal{X}_n$. Essentially, we have three requirements on $\mathcal{F}_n$: First, the number of functions in $\mathcal{F}_n$ should be at most polynomial in $n$. Second, in the limit of $n \to \infty$ the class $\mathcal{F}_n$ should be rich enough to approximate any measurable partition of the underlying space. Third, in order to perform the optimization we need to be able to enumerate all members of this class, that is the function class $\mathcal{F}_n$ should be "constructive" in some sense. A convenient choice satisfying all those properties is the class of "nearest neighbor partitions". This class contains all functions which can be generated as follows. Fix a subset of $m \ll n$ "seed points" $X_{s_1}, \ldots, X_{s_m}$ among the given data points. Assign all other data points to their closest seed points, that is for all $j = 1, \ldots, m$ define the set $Z_j$ as the subset of data points whose nearest seed point is $X_{s_j}$. Then consider all partitions of $\mathcal{X}_n$ which are constant on the sets $Z_j$. More formally, for given seeds we define the set $\mathcal{F}_n$ as the set of all functions $f : \mathcal{X} \to \{1, \ldots, K\}$ which are constant on the cells of the Voronoi partition induced by the seeds. Here $K$ denotes the number of clusters we want to construct. The function class $\mathcal{F}_n$ contains $K^m$ functions, which is polynomial in $n$ if the number $m$ of seeds satisfies $m = O(\log n)$. Given $\mathcal{F}_n$, the simplest polynomial-time optimization algorithm is then to evaluate $Q_n(f)$ for all $f \in \mathcal{F}_n$ and choose the solution $f_n = \operatorname{argmin}_{f \in \mathcal{F}_n} Q_n(f)$. We call the resulting clustering the *nearest neighbor clustering* and denote it by $\mathrm{NNC}(Q_n)$. In practice, the seeds will be chosen randomly among the given data points.

# 3 Consistency of nearest neighbor clustering

In this section we prove that nearest neighbor clustering is statistically consistent for many clustering quality functions. Due to the complexity of the proofs and the page restriction we can only present sketches of the proofs. All details can be found in von Luxburg et al. (2007). Let us start with some notation. For any clustering function $f : \mathbb{R}^d \to \{1, \ldots, K\}$ we denote by the predicate $A(f)$ a property of the function which can either be true or false. As an example, define $A(f)$ to be true if all clusters have at least a certain minimal size. Moreover, we need to introduce a predicate $A_n(f)$ which will be an "estimator" of $A(f)$ based on the finite sample only. Let $m := m(n) \leq n$ be the number of seeds used in nearest neighbor clustering. To simplify notation we assume in this section that the seeds are the first $m$ data points; all results remain valid for any other (even random) choice of seeds. As data space we use $\mathcal{X} = \mathbb{R}^d$. We define:

$$\mathrm{NN}_m(x) := \mathrm{NN}_{m(n)}(x) := \mathrm{argmin}_{y \in \{X_1, \ldots, X_m\}} \|x - y\| \quad (\text{ for } x \in \mathbb{R}^d)$$

$$\mathcal{F} := \{f : \mathbb{R}^d \to \{1, \ldots, K\} \mid f \text{ continuous } \mathbb{P}\text{-a.e. and } A(f) \text{ true}\}$$

$$\mathcal{F}_n := \mathcal{F}_{X_1, \ldots, X_n} := \{f : \mathbb{R}^d \to \{1, \ldots, K\} \mid f \text{ satisfies } f(x) = f(\mathrm{NN}_m(x)), \text{ and } A_n(f) \text{ is true}\}$$

$$\widetilde{\mathcal{F}}_n := \bigcup_{X_1, \ldots, X_n \in \mathbb{R}^d} \mathcal{F}_{X_1, \ldots, X_n}$$

Furthermore, let $Q : \mathcal{F} \to \mathbb{R}$ be the quality function we aim to minimize, and $Q_n : \mathcal{F}_n \to \mathbb{R}$ an estimator of this quality function on a finite sample. With this notation, the true clustering $f^*$ on the underlying space and the nearest neighbor clustering $f_n$ introduced in the last section are given by

$$f^* \in \mathrm{argmin}_{f \in \mathcal{F}} Q(f) \qquad \text{and} \qquad f_n \in \mathrm{argmin}_{f \in \mathcal{F}_n} Q_n(f).$$

Later on we will also need to work with the functions

$$f_n^* \in \mathrm{argmin}_{f \in \mathcal{F}_n} Q(f) \qquad \text{and} \qquad \widetilde{f}^*(x) := f^*(\mathrm{NN}_m(x)).$$

As distance function between different clusterings $f, g$ we will use

$$\mathrm{L}_n(f, g) := \mathbb{P}(f(X) \neq g(X) \mid X_1, \ldots, X_n)$$

(we need the conditioning in case $f$ or $g$ depend on the data, it has no effect otherwise).

**Theorem 1 (Consistency of nearest neighbor clustering)** *Let $(X_i)_{i \in \mathbb{N}}$ be a sequence of points drawn i.i.d. according to some probability measure $\mathbb{P}$ on $\mathbb{R}^d$, and $m := m(n)$ the number of seed points used in nearest neighbor clustering. Let $Q : \mathcal{F} \to \mathbb{R}$ be a clustering quality function, $Q_n : \widetilde{\mathcal{F}}_n \to \mathbb{R}$ its estimator, and $A(f)$ and $A_n(f)$ some predicates. Assume that:*

*1. $Q_n(f)$ is a consistent estimator of $Q(f)$ which converges sufficiently fast:*

$$\forall \varepsilon > 0, \quad K^m (2n)^{(d+1)m^2} \sup_{f \in \widetilde{\mathcal{F}}_n} \mathbb{P}(|Q_n(f) - Q(f)| > \varepsilon) \to 0.$$

*2. $A_n(f)$ is an estimator of $A(f)$ which is "consistent" in the following way:*

$$\mathbb{P}(A_n(\widetilde{f}^*) \text{ true}) \to 1 \qquad \text{and} \qquad \mathbb{P}(A(f_n) \text{ true}) \to 1.$$

*3. $Q$ is uniformly continuous with respect to the distance $L_n$ between $\mathcal{F}$ and $\mathcal{F}_n$:*

$$\forall \varepsilon > 0 \, \exists \delta(\varepsilon) > 0 \, \forall f \in \mathcal{F} \, \forall g \in \mathcal{F}_n : \quad \mathrm{L}_n(f, g) \leq \delta(\varepsilon) \implies |Q(f) - Q(g)| \leq \varepsilon.$$

*4. $\lim_{n \to \infty} m(n) = +\infty$.*

*Then nearest neighbor clustering as introduced in Section 2 is weakly consistent, that is $Q(f_n) \to Q(f^*)$ in probability.*

*Proof. (Sketch, for details see von Luxburg et al. (2007)).* We split the term $\mathbb{P}(|Q(f_n) - Q(f^*)| \geq \varepsilon)$ into its two sides $\mathbb{P}(Q(f_n) - Q(f^*) \leq -\varepsilon)$ and $\mathbb{P}(Q(f_n) - Q(f^*) \geq \varepsilon)$. It is a straightforward consequence of Condition (2) that the first term converges to 0. The main work consists in bounding the second term. As usual we consider the estimation and approximation errors

$$\mathbb{P}\big(Q(f_n) - Q(f^*) \geq \varepsilon\big) \leq \mathbb{P}\big(Q(f_n) - Q(f_n^*) \geq \varepsilon/2\big) + \mathbb{P}\big(Q(f_n^*) - Q(f^*) \geq \varepsilon/2\big).$$

First we bound the estimation error. In a few lines one can show that
$$\mathbb{P}(Q(f_n) - Q(f_n^*) \geq \varepsilon/2) \;\leq\; \mathbb{P}(\sup_{f\in\mathcal{F}_n} |Q_n(f) - Q(f)| \geq \varepsilon/4).$$
Note that even though the right hand side resembles the standard quantities often considered in statistical learning theory, it is not straightforward to bound as we do not assume that $Q(f) = \mathbb{E}Q_n(f)$. Moreover, note that the function class $\mathcal{F}_n$ is data dependent as the seed points used in the Voronoi partition are data points. To circumvent this problem, we replace the function class $\mathcal{F}_n$ by the larger class $\widetilde{\mathcal{F}}_n$, which is not data dependent. Using symmetrization by a ghost sample (cf. Section 12.3 of Devroye et al., 1996), we then move the supremum out of the probability:

$$\mathbb{P}\Big(\sup_{f\in\mathcal{F}_n} |Q_n(f) - Q(f)| \geq \varepsilon/4\Big) \;\leq\; 2S_K(\widetilde{\mathcal{F}}_n, 2n) \frac{\sup_{f\in\widetilde{\mathcal{F}}_n} \mathbb{P}\big(|Q_n(f) - Q(f)| \geq \varepsilon/16\big)}{\inf_{f\in\widetilde{\mathcal{F}}_n} \mathbb{P}\big(|Q_n(f) - Q(f)| \leq \varepsilon/8\big)} \quad (1)$$

Note that the unusual denominator in Eq. (1) emerges in the symmetrization step as we do not assume $Q(f) = \mathbb{E}Q_n(f)$. The quantity $S_K(\widetilde{\mathcal{F}}_n, 2n)$ denotes the shattering coefficient, that is the maximum number of ways that $2n$ points can be partitioned into $K$ sets using the functions in $\widetilde{\mathcal{F}}_n$. It is well known (e.g., Section 21.5 of Devroye et al., 1996) that the number of Voronoi partitions of $n$ points using $m$ cells in $\mathbb{R}^d$ is bounded by $n^{(d+1)m^2}$, hence the number of nearest neighbor clusterings into $K$ classes is bounded by $S_K(\widetilde{\mathcal{F}}_n, n) \leq K^m n^{(d+1)m^2}$. Under Condition (1) of the Theorem we now see that for fixed $\varepsilon$ and $n \to \infty$ the right hand side of (1) converges to 0. Thus the same holds for the estimation error. To deal with the approximation error, observe that if $A_n(\widetilde{f}^*)$ is true, then $\widetilde{f}^* \in \mathcal{F}_n$, and by the definition of $f_n^*$ we have

$$Q(f_n^*) - Q(f^*) \leq Q(\widetilde{f}^*) - Q(f^*) \quad \text{and thus}$$

$$\mathbb{P}\big(Q(f_n^*) - Q(f^*) \geq \varepsilon\big) \;\leq\; \mathbb{P}\big(A_n(\widetilde{f}^*) \text{ false}\big) \;+\; \mathbb{P}\big(\widetilde{f}^* \in \mathcal{F}_n \text{ and } Q(\widetilde{f}^*) - Q(f^*) \geq \varepsilon\big). \quad (2)$$

The first expression on the right hand side converges to 0 by Condition (2) in the theorem. Using Condition (3), we can bound the second expression in terms of the distance $L_n$ to obtain

$$\mathbb{P}\big(\widetilde{f}^* \in \mathcal{F}_n, \; Q(\widetilde{f}^*) - Q(f^*) \geq \varepsilon\big) \;\leq\; \mathbb{P}\big(Q(\widetilde{f}^*) - Q(f^*) \geq \varepsilon\big) \;\leq\; \mathbb{P}\big(\mathrm{L}_n(f^*, \widetilde{f}^*) \geq \delta(\varepsilon)\big).$$

Now we use techniques from Fritz (1975) to show that if $n$ is large enough, then the distance between a function $f \in \mathcal{F}$ evaluated at $x$ and the same function evaluated at $\mathrm{NN}_m(x)$ is small. Namely, for any $f \in \mathcal{F}$ and any $\varepsilon > 0$ there exists some $b(\delta(\varepsilon)) > 0$ which does not depend on $n$ and $f$ such that

$$\mathbb{P}(\mathrm{L}_n(f, f(\mathrm{NN}_m(\cdot))) > \delta(\varepsilon)) \;\leq\; (2/\delta(\varepsilon))\, e^{-mb(\delta(\varepsilon))}.$$

The quantity $\delta(\varepsilon)$ has been introduced in Condition (3). For every fixed $\varepsilon$, this term converges to 0 due to Condition (4), thus the approximation error vanishes. ☺

Now we want to apply our general theorem to particular objective functions. We start with the normalized cut. Let $s : \mathbb{R}^d \times \mathbb{R}^d \to \mathbb{R}^+$ be a similarity function which is upper bounded by a constant $C$. For a clustering $f : \mathbb{R}^d \to \{1, \ldots, K\}$ denote by $f_k(x) := \mathbb{1}_{f(x)=k}$ the indicator function of the $k$-th cluster. Define the empirical and true cut, volume, and normalized cut as follows:

$$\mathrm{cut}_n(f_k) := \frac{1}{n(n-1)} \sum_{i,j=1}^{n} f_k(X_i)(1 - f_k(X_j)) s(X_i, X_j)$$

$$\mathrm{cut}(f_k) := \mathbb{E}_{X,Y}\big(f_k(X)(1 - f_k(Y)) s(X, Y)\big)$$

$$\mathrm{vol}_n(f_k) := \frac{1}{n(n-1)} \sum_{i,j=1}^{n} f_k(X_i) s(X_i, X_j) \qquad\qquad \mathrm{vol}(f_k) := \mathbb{E}_{X,Y}\big(f_k(X) s(X, Y)\big)$$

$$\mathrm{Ncut}_n(f) := \sum_{k=1}^{K} \frac{\mathrm{cut}_n(f_k)}{\mathrm{vol}_n(f_k)} \qquad\qquad\qquad\qquad \mathrm{Ncut}(f) := \sum_{k=1}^{K} \frac{\mathrm{cut}(f_k)}{\mathrm{vol}(f_k)}$$

Note that $\mathbb{E}\,\mathrm{Ncut}_n(f) \neq \mathrm{Ncut}(f)$, but $\mathbb{E}\,\mathrm{cut}_n(f) = \mathrm{cut}(f)$ and $\mathbb{E}\,\mathrm{vol}_n(f) = \mathrm{vol}(f)$. We fix a constant $a > 0$, a sequence $(a_n)_{n\in\mathbb{N}}$ with $a_n \geq a_{n+1}$ and $a_n \to a$ and define the predicates

$$A(f) \text{ is true } :\Longleftrightarrow\; \mathrm{vol}(f_k) > a \;\; \forall k = 1, \ldots, K$$
$$A_n(f) \text{ is true } :\Longleftrightarrow\; \mathrm{vol}_n(f_k) > a_n \;\; \forall k = 1, \ldots, K \qquad\qquad (3)$$

**Theorem 2 (Consistency of NNC(Ncut$_n$))** *Let $(X_i)_{i\in\mathbb{N}}$ be a sequence of points drawn i.i.d. according to some probability measure $\mathbb{P}$ on $\mathbb{R}^d$ and $s : \mathbb{R}^d \times \mathbb{R}^d \to \mathbb{R}^+$ be a similarity function which is upper bounded by a constant $C$. Let $m := m(n)$ be the number of seed points used in nearest neighbor clustering, $a > 0$ an arbitrary constant, and $(a_n)_{n\in\mathbb{N}}$ a monotonically decreasing sequence with $a_n \to a$. Then nearest neighbor clustering using $Q := \mathrm{Ncut}$, $Q_n := \mathrm{Ncut}_n$, and $A$ and $A_n$ as defined in (3) is weakly consistent if $m(n) \to \infty$ and $m^2 \log n / (n(a - a_n)^2) \to 0$.*

*Proof.* We will check that all conditions of Theorem 1 are satisfied. First we establish that

$$\{|\operatorname{cut}_n(f_k) - \operatorname{cut}(f_k)| \leq a\varepsilon\} \cap \{|\operatorname{vol}_n(f_k) - \operatorname{vol}(f_k)| \leq a\varepsilon\} \subset \{|\frac{\operatorname{cut}_n(f_k)}{\operatorname{vol}_n(f_k)} - \frac{\operatorname{cut}(f_k)}{\operatorname{vol}(f_k)}| \leq 2\varepsilon\}.$$

Applying the McDiarmid inequality to $\operatorname{cut}_n$ and $\operatorname{vol}_n$, respectively, we obtain that for all $f \in \widetilde{\mathcal{F}}_n$

$$\mathbb{P}(|\operatorname{Ncut}(f) - \operatorname{Ncut}_n(f)| > \varepsilon) \leq 4K \exp\left(-\frac{na^2\varepsilon^2}{8C^2K^2}\right).$$

Together with $m^2 \log n/(n(a - a_n)^2) \to 0$ this shows Condition (1) of Theorem 1. The proof of Condition (2) is rather technical, but in the end also follows by applying the McDiarmid inequality to $\operatorname{vol}_n(f)$. Condition (3) follows by establishing that for $f \in \mathcal{F}$ and $g \in \mathcal{F}_n$ we have

$$|\operatorname{Ncut}(f) - \operatorname{Ncut}(g)| \leq \frac{4CK}{a} L_n(f, g).$$

<div align="right">☺</div>

In fact, Theorem 1 can be applied to a large variety of clustering objective functions. As examples, consider ratio cut, within-sum of squares, and the ratio of between- and within-cluster similarity:

$$\operatorname{RatioCut}_n(f) := \sum_{k=1}^K \frac{\operatorname{cut}_n(f_k)}{n_k} \qquad\qquad \operatorname{RatioCut}(f) := \sum_{k=1}^K \frac{\operatorname{cut}(f_k)}{\mathbb{E}f_k(X)}$$

$$\operatorname{WSS}_n(f) := \frac{1}{n}\sum_{i=1}^n \sum_{k=1}^K f_k(X_i)\|X_i - c_{k,n}\|^2 \qquad \operatorname{WSS}(f) := \mathbb{E}\sum_{k=1}^K f_k(X)\|X - c_k\|^2$$

$$\operatorname{BW}_n := \sum_{k=1}^K \frac{\operatorname{cut}_n(f_k)}{\operatorname{vol}_n(f_k) - \operatorname{cut}_n(f_k)} \qquad \operatorname{BW} := \sum_{k=1}^K \frac{\operatorname{cut}(f_k)}{\operatorname{vol}(f_k) - \operatorname{cut}(f_k)}$$

Here $n_k := \sum_i f_k(X_i)/n$ is the fraction of points in the $k$-th cluster, and $c_{k,n} := \sum_i f_k(X_i)X_i/(nn_k)$ and $c_k := \mathbb{E}f_k(X)X/\mathbb{E}f_k(X)$ are the empirical and true cluster centers.

**Theorem 3 (Consistency of $\mathbf{NNC(RatioCut_n)}$, $\mathbf{NNC(WSS_n)}$, and $\mathbf{NNC(BW_n)}$)** *Let $f_n$ and $f^*$ be the empirical and true minimizers of nearest neighbor clustering using $\operatorname{RatioCut}_n$, $\operatorname{WSS}_n$, or $\operatorname{BW}_n$, respectively. Then, under conditions similar to the ones in Theorem 2, we have $\operatorname{RatioCut}(f_n) \to \operatorname{RatioCut}(f^*)$, $\operatorname{WSS}(f_n) \to \operatorname{WSS}(f^*)$, and $\operatorname{BW}(f_n) \to \operatorname{BW}(f^*)$ in probability. See von Luxburg et al. (2007) for details.*

## 4 Implementation using branch and bound

It is an obvious question how nearest neighbor clustering can be implemented in a more efficient way than simply trying all functions in $\mathcal{F}_n$. Promising candidates are branch and bound methods. They are guaranteed to achieve an optimal solution, but in most cases are much more efficient than a naive implementation. As an example we introduce a branch and bound algorithm for solving $\operatorname{NNC(Ncut)}$ for $K = 2$ clusters. For background reading see Brusco and Stahl (2005). First of all, observe that minimizing $\operatorname{Ncut}_n$ over the nearest neighbor function set $\mathcal{F}_n$ is the same as minimizing $\operatorname{Ncut}_m$ over all partitions of a contracted data set consisting of $m$ "super-points" $Z_1, \ldots, Z_m$ (super-point $Z_i$ contains all data points assigned to the $i$-th seed point), endowed with the "super-similarity" function $\bar{s}(Z_s, Z_t) := \sum_{X_i \in Z_s, X_j \in Z_t} s(X_i, X_j)$. Hence nearest neighbor clustering on the original data set with $n$ points can be performed by directly optimizing Ncut on the contracted data set consisting of only $m$ super-points. Assume we already determined the labels $l_1, \ldots, l_{i-1} \in \{\pm 1\}$ of the first $i-1$ super-points. For those points we introduce the sets $A = \{Z_1, \ldots, Z_{i-1}\}$, $A^- := \{Z_j \mid j < i, l_j = -1\}$, $A^+ := \{Z_j \mid j < i, l_j = +1\}$, for the remaining points the set $B = \{Z_i, \ldots, Z_m\}$, and the set $V := A \cup B$ of all points. By default we label all points in $B$ with $-1$ and, in recursion level $i$, decide about moving $Z_i$ to cluster $+1$. Analogously to the notation $f_k$ of the previous section, in case $K = 2$ we can decompose $\operatorname{Ncut}(f) = \operatorname{cut}(f_{+1}) \cdot (1/\operatorname{vol}(f_{+1}) + 1/\operatorname{vol}(f_{-1}))$; we call the first term the "cut term" and the second term the "volume term". As it is standard in branch and bound we have to investigate whether the "branch" of clusterings with the specific fixed labels on $A$ could contain a solution which is better than all the previously considered solutions. We use two criteria for this purpose. The first one is very simple: assigning at least one point in $B$ to $+1$ can only lead to an improvement if this either decreases the cut term or the volume term of Ncut. Necessary conditions for this are $\max_{j \geq i} \bar{s}(Z_j, A^+) - \bar{s}(Z_j, A^-) \geq 0$ or $\operatorname{vol}(A^+) \leq \operatorname{vol}(V)/2$, respectively. If neither is satisfied, we retract. The second criterion involves a lower bound $\theta_l$ on the Ncut value of

```
Branch and bound algorithm for Ncut: f* = bbncut(S̄, i, f, θᵤ){
    1. Set g := f; set A⁻, A⁺, and B as described in the text
    2. // Deal with special cases:
        • If i = m and A⁻ = ∅ then return f.
        • If i = m and A⁻ ≠ ∅:
            – Set gᵢ = +1.
            – If Ncut(g) < Ncut(f) return g, else return f.
    3. // Pruning:
        • If vol(A⁺) > vol(A ∪ B)/2 and maxⱼ≥ᵢ(s̄(j, A⁺) − s̄(j, A⁻)) ≤ 0 return f.
        • Compute lower bound θₗ as described in the text.
        • If θₗ ≥ θᵤ then return f.
    4. // If no pruning possible, recursively call bbncut:
        • Set gᵢ = +1, θ'ᵤ := min{Ncut(g), θᵤ}, call g' := bbncut(S̄, g, i+1, θ'ᵤ)
        • Set gᵢ = −1, θ''ᵤ := min{Ncut(g'), θ'ᵤ}, call g'' := bbncut(S̄, g, i+1, θ''ᵤ)
        • If Ncut(g') ≤ Ncut(g'') then return g', else return g''.              }
```

Figure 1: *Branch and bound algorithm for* NNC(Ncut) *for* $K = 2$. *The algorithm is initially called with the super-similarity matrix* $\bar{S}$, $i = 2$, $f = (+1, -1, \ldots, -1)$, *and* $\theta_u$ *the Ncut value of* $f$.

all solutions in the current branch. It compares $\theta_l$ to an upper bound $\theta_u$ on the optimal Ncut value, namely to the Ncut value of the best function we have seen so far. If $\theta_l \geq \theta_u$ then no improvement is possible by any clustering in the current branch of the tree, and we retract. To compute $\theta_l$, assume we assign a non-empty set $B^+ \subset B$ to label +1 and the remaining set $B^- = B \setminus B^+$ to label -1. Using the conventions $\bar{s}(A, B) = \sum_{Z_i \in A, Z_j \in B} \bar{s}_{ij}$ and $\bar{s}(A, \emptyset) = 0$, the cut term is bounded by

$$\text{cut}(A^+ \cup B^+, A^- \cup B^-) \geq \begin{cases} \min_{j \geq i} s(Z_j, A^+) & \text{if } A^- = \emptyset \\ \bar{s}(A^+, A^-) + \min_{j \geq i} \bar{s}(Z_j, A^-) & \text{otherwise.} \end{cases} \quad (4)$$

The volume term can be maximally decreased in case $\text{vol}(A^+) < \text{vol}(V)/2$, when choosing $B^+$ such that $\text{vol}(A^+ \cup B^+) = \text{vol}(A^- \cup B^-) = \text{vol}(V)/2$. If $\text{vol}(A^+) > \text{vol}(V)/2$, then an increase of the volume term is unavoidable; this increase is minimal when we move one vertex only to $A^+$:

$$\frac{1}{\text{vol}(A^+ \cup B^+)} + \frac{1}{\text{vol}(A^- \cup B^-)} \geq \begin{cases} 4/\text{vol}(V) & \text{if } \text{vol}(A^+) \leq \text{vol}(V)/2 \\ \text{vol}(V)/\max_{j \geq i} \left(\text{vol}(A^+ \cup Z_j) \text{vol}(A^- \cup B \setminus Z_j)\right) & \text{otherw.} \end{cases} \quad (5)$$

Combining both bounds we can now define the lower bound $\theta_l$ as the product of Eq. (4) and (5). The entire algorithm is presented in Fig. 1. On top of the basic algorithm one can apply various heuristics to improve the retraction behavior and thus the average running time of the algorithm. For example, in our experience it is of advantage to sort the super-points by decreasing degree, and from one recursion level to the next one alternate between first visiting branch $g_i = 1$ and $g_i = -1$.

## 5 Experiments

The main point about nearest neighbor clustering is its statistical consistency: for large $n$ it reveals an approximately correct clustering. In this section we want to show that it also behaves reasonably on smaller samples. Given an objective function $Q_n$ (such as WSS or Ncut) we compare the NNC results to heuristics designed to optimize $Q_n$ directly (such as $k$-means or spectral clustering). As numeric data sets we used classification benchmark data sets from different repositories (UCI repository, repository by G. Rätsch) and microarray data from Spellman et al. (1998). Moreover, we use graph data sets of the internet graph and of biological, social, and political networks: COSIN collection, collection by M. Newman, email data by Guimerà et al. (2003), electrical power network by Watts and Strogatz (1998), and protein interaction networks of Jeong et al. (2001) and Tsuda et al. (2005). Due to space constraints we focus on the case of constructing $K = 2$ clusters using the objective functions WSS and Ncut. We always set the number $m$ of seed points for NNC to $m = \log n$. In case of WSS, we compare the result of the $k$-means algorithm to the result of NNC using the WSS objective function and the Euclidean distance to assign data points to seed points.

| Numeric data sets | WSS | | Ncut | |
|---|---|---|---|---|
| | $K$-means | NNC | SC | NNC |
| breast-c. | $6.95 \pm 0.19$ | $7.04 \pm 0.21$ | $0.11 \pm 0.02$ | $0.09 \pm 0.02$ |
| | $7.12 \pm 0.20$ | $7.12 \pm 0.22$ | $0.22 \pm 0.07$ | $0.21 \pm 0.07$ |
| diabetis | $6.62 \pm 0.22$ | $6.71 \pm 0.22$ | $0.03 \pm 0.02$ | $0.03 \pm 0.02$ |
| | $6.72 \pm 0.22$ | $6.72 \pm 0.22$ | $0.04 \pm 0.03$ | $0.05 \pm 0.05$ |
| german | $18.26 \pm 0.27$ | $18.56 \pm 0.28$ | $0.02 \pm 0.02$ | $0.02 \pm 0.02$ |
| | $18.35 \pm 0.30$ | $18.45 \pm 0.32$ | $0.04 \pm 0.08$ | $0.03 \pm 0.03$ |
| heart | $10.65 \pm 0.46$ | $10.77 \pm 0.47$ | $0.18 \pm 0.03$ | $0.17 \pm 0.02$ |
| | $10.75 \pm 0.46$ | $10.74 \pm 0.46$ | $0.28 \pm 0.03$ | $0.30 \pm 0.07$ |
| splice | $68.99 \pm 0.24$ | $69.89 \pm 0.24$ | $0.36 \pm 0.10$ | $0.44 \pm 0.16$ |
| | $69.03 \pm 0.24$ | $69.18 \pm 0.25$ | $0.58 \pm 0.09$ | $0.66 \pm 0.18$ |
| bcw | $3.97 \pm 0.26$ | $3.98 \pm 0.26$ | $0.02 \pm 0.01$ | $0.02 \pm 0.01$ |
| | $3.98 \pm 0.26$ | $3.98 \pm 0.26$ | $0.04 \pm 0.01$ | $0.08 \pm 0.07$ |
| ionosph. | $25.72 \pm 1.63$ | $25.77 \pm 1.63$ | $0.06 \pm 0.03$ | $0.04 \pm 0.01$ |
| | $25.76 \pm 1.63$ | $25.77 \pm 1.63$ | $0.12 \pm 0.11$ | $0.14 \pm 0.12$ |
| pima | $6.62 \pm 0.22$ | $6.73 \pm 0.23$ | $0.03 \pm 0.03$ | $0.03 \pm 0.03$ |
| | $6.73 \pm 0.23$ | $6.73 \pm 0.23$ | $0.05 \pm 0.04$ | $0.09 \pm 0.13$ |
| cellcycle | $0.78 \pm 0.03$ | $0.78 \pm 0.03$ | $0.12 \pm 0.02$ | $0.10 \pm 0.01$ |
| | $0.78 \pm 0.03$ | $0.78 \pm 0.02$ | $0.16 \pm 0.02$ | $0.15 \pm 0.03$ |

| Network data | NNC | SC |
|---|---|---|
| ecoli.interact | 0.06 | 0.06 |
| ecoli.metabol | 0.03 | 0.04 |
| helico | 0.16 | 0.16 |
| beta3s | 0.00 | 0.00 |
| AS-19971108 | 0.02 | 0.02 |
| AS-19980402 | 0.01 | 1.00 |
| AS-19980703 | 0.02 | 0.02 |
| AS-19981002 | 0.04 | 0.04 |
| AS-19990114 | 0.08 | 0.05 |
| AS-19990402 | 0.11 | 0.10 |
| netscience | 0.01 | 0.01 |
| polblogs | 0.11 | 0.11 |
| power | 0.00 | 0.00 |
| email | 0.27 | 0.27 |
| yeastProtInt | 0.04 | 0.06 |
| protNW1 | 0.00 | 0.00 |
| protNW2 | 0.08 | 1.00 |
| protNW3 | 0.01 | 0.80 |
| protNW4 | 0.03 | 0.76 |

Table 1: *Left: Numeric data.* Results for $K$-means algorithm, NNC(WSS) with Euclidean distance; spectral clustering (SC); NNC(Ncut) with commute distance. The top line always shows the results on the training set, the second line the extended results on the test set. *Right: Network data.* NNC(Ncut) with commute distance and spectral clustering, both trained on the entire graph.

Note that one cannot run $K$-means on pure network data, which does not provide coordinates. In case of Ncut, we use the Gaussian kernel as similarity function on the numeric data sets. The kernel width $\sigma$ is set to the mean distance of a data point to its $k$-th nearest neighbor. We then build the $k$-nearest neighbor graph (both times using $k = \ln n$). On the network data, we directly use the given graph. For both types of data, we use the commute distance on the graph (e.g., Gutman and Xiao, 2004) as distance function to determine the nearest seed points for NNC.

In the first experiment we compare the values obtained by the different algorithms on the training sets. From the numeric data sets we generated $z = 40$ training sets by subsampling $n/2$ points. On each training set, we repeated all algorithms $r = 50$ times with different random initializations (the seeds in NNC; the centers in $K$-means; the centers in the $K$-means post-processing step in spectral clustering). Denoting the quality of an individual run of the algorithm by $q$, we then report the values $mean_z(min_r q) \pm standarddev_z(min_r q)$. For the network data sets we ran spectral clustering and NNC on the whole graph. Again we use $r = 50$ different initializations, and we report $min_r q$. All results can be found in Table 1. For both the numeric data sets (left table, top lines) and the network data sets (right table) we see that the training performance of NNC is comparable to the other algorithms. This is what we had hoped, and we find it remarkable as NNC is in fact a very simple clustering algorithm.

In the second experiment we try to measure the amount of overfitting induced by the different algorithms. For each of the numeric data sets we cluster $n/2$ points, extend the clustering to the other $n/2$ points, and then compute the objective function on the test set. For the extensions we proceed in a greedy way: for each test point, we add this test point to the training set and then give it the label +1 or -1 that leads to the smaller quality value on the augmented training set. We also tried several other extensions suggested in the literature, but the results did not differ much. To compute the test error, we then evaluate the quality function on the test set labeled according to the extension. For Ncut, we do this based on the $k$-nearest neighbor graph on the test set only. Note that this experiment does not make sense on the network data, as there is no default procedure to construct the subgraphs for training and testing. The results on the numeric data sets are reported in Table 1 (left table, bottom lines). We see that NNC performs roughly comparably to the other algorithms. This is not really what we wanted to obtain, our hope was that NNC obtains better test values as it is less prone to overfitting. The most likely explanation is that both $K$-means and spectral clustering have already reasonably good extension properties. This can be due to the fact that as NNC, both algorithms consider only a certain subclass of all partitions: Voronoi partitions for $K$-means, and partitions induced by eigenvectors for spectral clustering. See below for more discussion.

# 6 Discussion

In this paper we investigate clustering algorithms which minimize quality functions. Our main point is that, as soon as we require statistical consistency, we have to work with "small" function classes $\mathcal{F}_n$. If we even choose $\mathcal{F}_n$ to be polynomial, then all problems due to NP hardness of discrete optimization problems formally disappear as the remaining optimization problems become inherently polynomial. From a practical point of view, the approach of using a restricted function class $\mathcal{F}_n$ can be seen as a more controlled way of simplifying NP hard optimization problems than the standard approaches of local optimization or relaxation. Carefully choosing the function class $\mathcal{F}_n$ such that overly complex target functions are excluded, we can guarantee to pick the best out of all remaining target functions. This strategy circumvents the problem that solutions of local optimization or relaxation heuristics can be arbitrarily far away from the optimal solution.

The generic clustering algorithm we studied in this article is nearest neighbor clustering, which produces clusterings that are constant on small local neighborhoods. We have proved that this algorithm is statistically consistent for a large variety of popular clustering objective functions. Thus, as opposed to other clustering algorithms such as the $K$-means algorithm or spectral clustering, nearest neighbor clustering is guaranteed to converge to a minimizer of the true global optimum on the underlying space. This statement is much stronger than the results already known for $K$-means or spectral clustering. For $K$-means it has been proved that the global minimizer of the WSS objective function on the sample converges to a global minimizer on the underlying space (e.g., Pollard, 1981). However, as the standard $K$-means algorithm only discovers a local optimum on the discrete sample, this result does not apply to the algorithm used in practice. A related effect happens for spectral clustering, which is a relaxation attempting to minimize Ncut (see von Luxburg (2007) for a tutorial). It has been shown that under certain conditions the solution of the relaxed problem on the finite sample converges to some limit clustering (e.g., von Luxburg et al., to appear). However, it has been conjectured that this limit clustering is not necessarily the optimizer of the Ncut objective function. So for both cases, our consistency results represent an improvement: our algorithm provably converges to the true limit minimizer of $K$-means or Ncut, respectively. The same result also holds for a large number of alternative objective functions used for clustering.

# References

M. Brusco and S. Stahl. *Branch-and-Bound Applications in Combinatorial Data Analysis*. Springer, 2005.

S. Bubeck and U. von Luxburg. Overfitting of clustering and how to avoid it. Preprint, 2007.

Data repository by G. Rätsch. http://ida.first.fraunhofer.de/projects/bench/benchmarks.htm.

Data repository by M. Newman. http://www-personal.umich.edu/~mejn/netdata/.

Data repository by UCI. http://www.ics.uci.edu/~mlearn/MLRepository.html.

Data repository COSIN. http://151.100.123.37/data.html.

L. Devroye, L. Györfi, and G. Lugosi. *A Probabilistic Theory of Pattern Recognition*. Springer, 1996.

J. Fritz. Distribution-free exponential error bound for nearest neighbor pattern classification. *IEEE Trans. Inf. Th.*, 21(5):552 – 557, 1975.

R. Guimerà, L. Danon, A. Díaz-Guilera, F. Giralt, and A. Arenas. Self-similar community structure in a network of human interactions. *Phys. Rev. E*, 68(6):065103, 2003.

I. Gutman and W. Xiao. Generalized inverse of the Laplacian matrix and some applications. *Bulletin de l'Academie Serbe des Sciences at des Arts (Cl. Math. Natur.)*, 129:15 – 23, 2004.

H. Jeong, S. Mason, A. Barabasi, and Z. Oltvai. Centrality and lethality of protein networks. *Nature*, 411: 41 – 42, 2001.

D. Pollard. Strong consistency of k-means clustering. *Annals of Statistics*, 9(1):135 – 140, 1981.

P. Spellman, G. Sherlock, M. Zhang, V. Iyer, M. Anders, M. Eisen, P. Brown, D. Botstein, and B. Futcher. Comprehensive identification of cell cycle-regulated genes of the yeast saccharomyces cerevisiae by microarray hybridization. *Mol Biol Cell*, 9(12):3273–97, 1998.

K. Tsuda, H. Shin, and B. Schölkopf. Fast protein classification with multiple networks. *Bioinformatics*, 21 (Supplement 1):ii59 – ii65, 2005.

V. Vapnik. *The Nature of Statistical Learning Theory*. Springer, 1995.

U. von Luxburg. A tutorial on spectral clustering. *Statistics and Computing*, 17(4), 2007.

U. von Luxburg, S. Bubeck, S. Jegelka, and M. Kaufmann. Supplementary material to "Consistent minimization of clustering objective functions", 2007. http://www.tuebingen.mpg.de/~ule.

U. von Luxburg, M. Belkin, and O. Bousquet. Consistency of spectral clustering. *Annals of Statistics*, to appear.

D. Watts and S. Strogatz. Collective dynamics of small world networks. *Nature*, 393:440–442, 1998.

